# Smooth-projected Neighborhood Pursuit for High-dimensional Nonparanormal Graph Estimation

**Tuo Zhao**
Department of Computer Science
Johns Hopkins University

**Kathryn Roeder**
Department of Statistics
Carnegie Mellon University

**Han Liu**
Department of Operations Research and Financial Engineering
Princeton University

## Abstract

We introduce a new learning algorithm, named smooth-projected neighborhood pursuit, for estimating high dimensional undirected graphs. In particularly, we focus on the nonparanormal graphical model and provide theoretical guarantees for graph estimation consistency. In addition to new computational and theoretical analysis, we also provide an alternative view to analyze the tradeoff between computational efficiency and statistical error under a smoothing optimization framework. Numerical results on both synthetic and real datasets are provided to support our theory.

## 1 Introduction

We consider the undirected graph estimation problem for a $d$-dimensional random vector $\boldsymbol{X} = (X_1, ..., X_d)^T$ (Lauritzen, 1996; Wille et al., 2004; Blei and Lafferty, 2007; Honorio et al., 2009). More specifically, let $V$ be the set that contains nodes representing the $d$ variables in $\boldsymbol{X}$, and $E$ be the set that contains edges representing the conditional independence relationship among $X_1, ..., X_d$, we say that the distribution of $\boldsymbol{X}$ is Markov to $\mathcal{G} = (V, E)$ if $X_i$ is independent of $X_j$ given $\boldsymbol{X}_{\backslash\{i,j\}}$ for all $(i, j) \notin E$, where $\boldsymbol{X}_{\backslash\{i,j\}} = \{X_k : \ k \neq i, j\}$. Our goal is to recover $\mathcal{G}$ based on $n$ independent observations of $\boldsymbol{X}$.

Most existing methods for high dimensional graph estimation assume that the random vector $\boldsymbol{X}$ follows a Gaussian distribution, i.e., $\boldsymbol{X} \sim N(\boldsymbol{\mu}, \boldsymbol{\Sigma})$. Under this parametric assumption, the graph estimation problem can be solved by estimating the sparsity pattern of the precision matrix $\boldsymbol{\Omega} = \boldsymbol{\Sigma}^{-1}$, i.e., the nodes $i$ and $j$ are connected if and only if $\boldsymbol{\Omega}_{ij} \neq 0$. The problem of estimating the sparsity pattern of $\boldsymbol{\Omega}$ is also called covariance selection in Dempster (1972). There are two major approaches for learning high dimensional Gaussian graphical models: (i) graphical lasso (Yuan and Lin, 2007; Friedman et al., 2007; Banerjee et al., 2008) and (ii) neighborhood pursuit (Meinshausen and Bühlmann, 2006). The graphical lasso maximizes the $\ell_1$-penalized Gaussian likelihood and simultaneously estimates the precision matrix $\boldsymbol{\Omega}$ and graph $\mathcal{G}$. In contrast, the neighborhood pursuit method maximizes the $\ell_1$-penalized pseudo-likelihood and can only estimate the graph $\mathcal{G}$. Scalable software packages such as `glasso` and `huge` have been developed to implement these algorithms (Friedman et al., 2007; Zhao et al., 2012). Theoretically, both methods are consistent in graph recovery for Gaussian models under certain regularity conditions. However, Ravikumar et al. (2011) suspect that the neighborhood pursuit approach has a better sample complexity in graph recovery than the graphical lasso. Moreover, these two methods are often observed to behave differently on real datasets in practical applications.

In Liu et al. (2009), an semiparametric nonparanormal model is proposed to relax the restrictive normality assumption. More specifically, they assume that there exists a set of strictly monotone trans-

formations $f = (f_j)_{j=1}^d$, such that the transformed random vector $f(\boldsymbol{X}) = (f_1(X_1), \ldots, f_d(X_d))^T$ follows a Gaussian distribution, i.e., $f(\boldsymbol{X}) \sim N(\mathbf{0}, \boldsymbol{\Omega}^{-1})$. Liu et al. (2009) show that for the non-paranormal distribution, the graph $\mathcal{G}$ can also be estimated by examining the sparsity pattern of $\boldsymbol{\Omega}$. Different methods have been proposed to infer the nonparanormal model in high dimensions. In Liu et al. (2012), a rank-based estimator named nonparanormal SKEPTIC is proposed to directly estimate $\boldsymbol{\Omega}$. Their main idea is to calculate a rank-correlation matrix (either based on the Spearman's rho or Kendall's tau correlation) and plug the estimated correlation matrix into the graphical lasso to estimate $\boldsymbol{\Omega}$ and graph $\mathcal{G}$. Such a procedure has been proven to be robust and achieve the same parametric rates of convergence as the graphical lasso (Liu et al., 2012). However, how to combine the nonparanormal SKEPTIC estimator with the neighborhood pursuit approach is still an open problem. The main challenge is that the possible indefiniteness of the rank-based correlation matrix estimates could lead to a non-convex computational formulation. Such potential non-convexity challenges both computational and theoretical analysis.

In this paper, we bridge this gap by proposing a novel smooth-projected neighborhood pursuit method. The main idea is to project the possibly indefinite nonparanormal SKEPTIC correlation matrix estimator into the cone of all positive semi-definite matrices with respect to a smoothed elementwise $\ell_\infty$-norm. Such a projection step is closely related to the dual smoothing approach in Nesterov (2005). We provide both computational and theoretical analysis of the derived algorithm. Computationally, our proposed smoothed elementwise $\ell_\infty$-norm has nice structure so that we can develop an efficient fast proximal gradient solver with a provable convergence rate $O(1/\sqrt{\epsilon})$ ($\epsilon$ is the desired accuracy of the objective value, Nesterov (1988)). Theoretically, we provide sufficient conditions to guarantee that the proposed smooth-projected neighborhood pursuit approach is graph estimation consistent.

In addition to new computational and statistical analysis, we further provide an alternative view to analyze the fundamental tradeoff between computational efficiency and statistical error under the smoothing optimization framework. Existing literature (Nesterov, 2005; Chen et al., 2012) considers the dual smoothing approach as a tradeoff between computational efficiency and approximation error. To avoid a large approximation error, they need to restrict the smoothness and obtain a slower rate ($O(1/\epsilon)$ vs. $O(1/\sqrt{\epsilon})$). However, we directly consider the statistical error introduced by the smoothing approach, and show that the obtained estimator preserves the good statistical properties without losing the computational efficiency. Thus we get the good sides of both worlds.

The rest of this paper is organized as follows: The next section reviews the *nonparanormal* SKEPTIC in Liu et al. (2012); Section 3 introduces the smooth-projected neighborhood pursuit and derives the fast proximal gradient algorithm; Section 4 explores the statistical properties of the procedure; Section 5 and 6 present results on on both simulated and real datasets. Due to the space limit, most of technical details are put in a significantly extended version of this paper (Zhao et al., 2013). In addition, Zhao et al. (2013) also contains more thorough numerical experiments and detailed comparison with other competitors.

## 2 Background

We first introduce some notation. Let $\boldsymbol{v} = (v_1, \ldots, v_d)^T \in \mathbb{R}^d$, we define the vector norms: $||\boldsymbol{v}||_1 = \sum_j |v_j|$, $||\boldsymbol{v}||_2^2 = \sum_j v_j^2$, and $||\boldsymbol{v}||_\infty = \max_j |v_i|$. Let $\mathbf{A} = [\mathbf{A}_{jk}] \in \mathbb{R}^{d \times d}$ and $\mathbf{B} = [\mathbf{B}_{jk}] \in \mathbb{R}^{d \times d}$ be two symmetric matrices, we define the matrix operator norms: $||\mathbf{A}||_1 = \max_k \sum_j |\mathbf{A}_{jk}|$, $||\mathbf{A}||_\infty = \max_j \sum_k |\mathbf{A}_{jk}|$, $||\mathbf{A}||_2 = \max_{||\boldsymbol{v}||_2=1} ||\mathbf{A}\boldsymbol{v}||_2$ and elementwise norms $\|\mathbf{A}\|_1 = \sum_{j,k} |\mathbf{A}_{jk}|$, $\|\mathbf{A}\|_\infty = \max_{j,k} |\mathbf{A}_{jk}|$, $||\mathbf{A}||_\mathsf{F}^2 = \sum_{j,k} |\mathbf{A}_{jk}|^2$. We denote $\Lambda_{\min}(\mathbf{A})$ and $\Lambda_{\max}(\mathbf{A})$ as the smallest and largest eigenvalues of $\mathbf{A}$. The inner product of $\mathbf{A}$ and $\mathbf{B}$ is denoted by $\langle \mathbf{A}, \mathbf{B} \rangle = \mathrm{tr}(\mathbf{A}^T\mathbf{B})$, where $\mathrm{tr}(\cdot)$ is the trace operator.

We denote the subvector of $\boldsymbol{v}$ with the $j^{\text{th}}$ entry removed by $\boldsymbol{v}_{\backslash j} = (v_1, \ldots, v_{j-1}, v_{j+1}, \ldots, v_d)^T \in \mathbb{R}^{d-1}$. In a similar notion, we denote the $i^{\text{th}}$ row of $\mathbf{A}$ with its $j^{\text{th}}$ entry removed by $\mathbf{A}_{i,\backslash j}$. If $I$ is a set of indices, then the sub-matrix of $\mathbf{A}$ with both column and row indices in $I$ is denoted by $\mathbf{A}_{II}$.

We then introduce the nonparanormal graphical model. The nonparanormal (nonparametric normal) distribution was initially motivated by the sparse additive models (Ravikumar et al., 2009). It aims at separately modeling the marginal distribution and conditional independence structure. The formal definition is as follows,

**Definition 2.1** (Nonparanormal Distribution Liu et al. (2009)). *Let $f = \{f_1, ..., f_d\}$ be a collection of non-decreasing univariate functions and $\mathbf{\Sigma}^* \in \mathbb{R}^{d \times d}$ be a correlation matrix with $\mathrm{diag}(\mathbf{\Sigma}^*) = 1$. We say a d-dimensional random variable $\mathbf{X} = (X_1, ..., X_d)^T$ follows a nonparanormal distribution, denoted by $\mathbf{X} \sim NPN_d(f, \mathbf{\Sigma}^*)$, if*

$$f(\mathbf{X}) = (f_1(X_1), ..., f_d(X_d))^T \sim N(\mathbf{0}, \mathbf{\Sigma}^*). \tag{2.1}$$

The nonparanormal family is equivalent to the Gaussian copula family for continuous distributions (Klaassen and Wellner, 1997; Tsukahara, 2005; Liu et al., 2009). Similar to the Gaussian graphical model, the nonparanormal graphical model also encodes the conditional independence graph by the sparsity pattern of the precision matrix $\mathbf{\Omega}^* = (\mathbf{\Sigma}^*)^{-1}$. More details can be found in (Liu et al., 2009).

Recently, Liu et al. (2012) propose a rank-based procedure, named *nonparanormal* SKEPTIC, for learning nonparanromal graphical models. More specifically, let $\boldsymbol{x}^1, ..., \boldsymbol{x}^n$ with $\boldsymbol{x}^i = (x_1^i, ..., x_d^i)^T$ be $n$ independent observations of $\boldsymbol{X}$, we define the Spearman's rho and Kendall's tau correlation coefficients as

$$\text{Spearman's rho}: \quad \widehat{\rho}_{jk} = \frac{\sum_{i=1}^{n}(r_j^i - \bar{r}_j)(r_k^i - \bar{r}_k)}{\sqrt{\sum_{i=1}^{n}(r_j^i - \bar{r}_j)^2 \cdot \sum_{i=1}^{n}(r_k^i - \bar{r}_k)^2}}, \tag{2.2}$$

$$\text{Kendall's tau}: \quad \widehat{\tau}_{jk} = \frac{2}{n(n-1)} \sum_{i < i'} \mathrm{sign}\left(x_j^i - x_j^{i'}\right)\left(x_k^i - x_k^{i'}\right), \tag{2.3}$$

where $r_j^i$ denotes the rank of $x_j^i$ among $x_j^1, ..., x_j^n$ and $\bar{r}_j = \frac{1}{n}\sum_{i=1}^{n} r_j^i = (n+1)/2$. Both the Spearman's rho and Kendall's tau correlations are rank-based and invariant to univariate monotone transformations. The nonparanormal SKEPTIC estimators are defined as $\widehat{\mathbf{S}}^\rho = [\widehat{\mathbf{S}}_{jk}^\rho] \in \mathbb{R}^{d \times d}$ and $\widehat{\mathbf{S}}^\tau = [\widehat{\mathbf{S}}_{jk}^\tau] \in \mathbb{R}^{d \times d}$ calculated from

$$\widehat{\mathbf{S}}_{jk}^\rho = 2\sin\left(\frac{\pi}{6}\widehat{\rho}_{jk}\right) \quad \text{and} \quad \widehat{\mathbf{S}}_{jk}^\tau = \sin\left(\frac{\pi}{2}\widehat{\tau}_{jk}\right). \tag{2.4}$$

Here the $\sin(\cdot)$ transformations correct the population bias. $\widehat{\mathbf{S}}^\rho$ and $\widehat{\mathbf{S}}^\tau$ avoid explicitly calculating the marginal transformation functions $\{f_j\}_{j=1}^d$ and has been shown to achieve the optimal parametric rates of convergence (Liu et al., 2012). Since Liu et al. (2012) suggest that $\widehat{\mathbf{S}}^\rho$ and $\widehat{\mathbf{S}}^\tau$ have very similar performance, for notational simplicity, we simply omit the superscript ($\tau$ and $\rho$), and use $\widehat{\mathbf{S}}$ instead. Theoretically, Liu et al. (2012) establish the following concentration bound of the nonparanormal SKEPTIC estimator, which is a sufficient condition to achieve graph estimation consistency in high dimensions.

**Lemma 2.2** (Nonparanormal SKEPTIC, Liu et al. (2012)). *Given the nonparanormal SKEPTIC estimator $\widehat{\mathbf{S}}$, for large enough $n$, we have $\widehat{\mathbf{S}}$ satisfying*

$$\mathbb{P}\left(\|\widehat{\mathbf{S}} - \mathbf{\Sigma}^*\|_\infty \le 8\pi\varphi\right) \ge 1 - d^2\exp(-n\varphi^2). \tag{2.5}$$

In the next section we will introduce our new smooth-projected neighborhood pursuit method and show that it also admits a similar concentration bound.

## 3 Smooth-Projected Neighborhood Pursuit

Similar to the neighborhood pursuit, our smooth-projected neighborhood pursuit also solves a collection of $\ell_1$-penalized least square problems as follows,

$$\widehat{\mathbf{B}}_{\backslash j, j} = \underset{\mathbf{B}_{j,j}=0}{\mathrm{argmin}} \, \mathbf{B}_{\backslash j,j}^T \widetilde{\mathbf{S}}_{\backslash j, \backslash j} \mathbf{B}_{\backslash j,j} - 2\widetilde{\mathbf{S}}_{\backslash j,j}^T \mathbf{B}_{\backslash j,j} + \lambda\|\mathbf{B}_{\backslash j,j}\|_1 \text{ for all } j = 1, ..., d, \tag{3.1}$$

where $\widetilde{\mathbf{S}}$ is a positive semi-definite replacement of the nonparanormal SKEPTIC estimator $\widehat{\mathbf{S}}$. (3.1) can be efficiently solved by existing solvers such as the coordinate descent algorithm (Friedman et al., 2007). Let $I_j$ denote a set of vertices, that are the neighbors of of node $j$, and $J_j$ denote a set of vertices, that are not, then we obtain $\widehat{I}_j = \{k : \widehat{\mathbf{B}}_{jk} \ne 0\}$ and $\widehat{J}_j = \{k : \widehat{\mathbf{B}}_{jk} = 0\}$. Thus we can eventually get the graph estimator $\widehat{\mathcal{G}}$ by combining all $\widehat{I}_j$'s.

## 3.1 Smoothed Elementwise $\ell_\infty$-norm

Our proposed method starts with the following projection problem,

$$\overline{\mathbf{S}} = \underset{\mathbf{S}}{\operatorname{argmin}} \|\widehat{\mathbf{S}} - \mathbf{S}\|_\infty \ \text{ s.t. } \ \mathbf{S} \succeq 0. \tag{3.2}$$

From the triangle inequality and the fact that $\mathbf{\Sigma}^*$ is a feasible solution to (3.2), we have

$$\|\mathbf{\Sigma}^* - \widehat{\mathbf{S}} + \widehat{\mathbf{S}} - \overline{\mathbf{S}}\|_\infty \le \|\widehat{\mathbf{S}} - \overline{\mathbf{S}}\|_\infty + \|\widehat{\mathbf{S}} - \mathbf{\Sigma}^*\|_\infty \le 2\|\widehat{\mathbf{S}} - \mathbf{\Sigma}^*\|_\infty. \tag{3.3}$$

Then by combining Lemma 2.2 and (3.3), we can show that $\overline{\mathbf{S}}$ concentrates to $\mathbf{\Sigma}^*$ with a rate similar to Lemma 2.2. However, (3.2) is computationally expensive due to the non-smooth elementwise $\ell_\infty$-norm. To overcome this challenge, we apply the dual smoothing approach in Nesterov (2005) to efficiently solve (3.2) with a controllable loss in accuracy. More specifically, for any matrix $\mathbf{A} \in \mathbb{R}^{d \times d}$, we exploit the Fenchel's dual representation of the elementwise $\ell_\infty$-norm to obtain its smooth surrogate as follows,

$$\|\mathbf{A}\|_\infty^\mu = \max_{\|\mathbf{U}\|_1 \le 1} \langle \mathbf{U}, \mathbf{A} \rangle - \frac{\mu}{2} \|\mathbf{U}\|_{\mathsf{F}}^2, \tag{3.4}$$

where $\mu > 0$ is the smoothing parameter, and the second term is the proximity function of $\mathbf{U}$. We call $\|\mathbf{A}\|_\infty^\mu$ smoothed elementwise $\ell_\infty$-norm. A closed form solution to (3.4) is characterized in the following lemma.

**Lemma 3.1.** *Equation* (3.4) *has a closed form solution,* $\widetilde{\mathbf{U}}$ *with*

$$\widetilde{\mathbf{U}}_{jk} = \operatorname{sign}(\mathbf{A}_{jk}) \cdot \max\left\{ \left| \frac{\mathbf{A}_{jk}}{\mu} \right| - \gamma, 0 \right\}, \tag{3.5}$$

*where $\gamma$ is the minimum non-negative constant such that $\|\widetilde{\mathbf{U}}\|_1 \le 1$.*

By utilizing a suitable pivotal quantity, we can efficiently obtain $\gamma$ with the expected computational complexity $O(d^2)$. More details of the algorithm can be found in Zhao et al. (2013). The smoothed elementwise $\ell_\infty$-norm is a smooth convex function. Let $\mathbf{A} = \widehat{\mathbf{S}} - \mathbf{S}$, and we can evaluate its gradient using (3.5) as follows,

$$\nabla \|\widehat{\mathbf{S}} - \mathbf{S}\|_\infty^\mu = \frac{\partial \|\widehat{\mathbf{S}} - \mathbf{S}\|_\infty^\mu}{\partial(\widehat{\mathbf{S}} - \mathbf{S})} \cdot \frac{\partial(\widehat{\mathbf{S}} - \mathbf{S})}{\partial \mathbf{S}} = -\widetilde{\mathbf{U}}. \tag{3.6}$$

Since $\widetilde{\mathbf{U}}$ is essentially a soft thresholding function, therefore it is continuous in $\mathbf{S}$ with the Lipchitz constant $\mu^{-1}$. In the next section, we will show that by considering the following alternative optimization problem

$$\widetilde{\mathbf{S}} = \underset{\mathbf{S}}{\operatorname{argmin}} \|\widehat{\mathbf{S}} - \mathbf{S}\|_\infty^\mu \ \text{ s.t. } \ \mathbf{S} \succeq 0, \tag{3.7}$$

we can also obtain a good correlation estimator without losing computational efficiency.

## 3.2 Fast Proximal Gradient Algorithm

Equation (3.7) has a minimum eigenvalue constraint, regarding which, we exploit Nesterov (1988) and derive the following fast proximal gradient algorithm. The main idea is to utilize the gradients in previous iterations to help find the descent direction for the current iteration, and eventually achieves a faster convergence rate than the ordinary projected gradient algorithm. In this algorithm, we need two sequences of auxiliary variables $\mathbf{M}^{(t)}$ and $\mathbf{W}^{(t)}$ with $\mathbf{M}^{(0)} = \mathbf{W}^{(0)} = \mathbf{S}^{(0)}$, and a sequence of weights $\theta_t = 2/(1+t)$ where $t = 0, 1, 2, \dots$.

Before we proceed with the proposed algorithm, we describe Lemma 3.2, which can solve the following projection problem

$$\Pi_+(\mathbf{A}) = \underset{\mathbf{B}}{\operatorname{argmin}} \|\mathbf{B} - \mathbf{A}\|_{\mathsf{F}}^2 \text{ s.t. } \mathbf{B} \succeq 0, \tag{3.8}$$

where $\mathbf{A} \in \mathbb{R}^{d \times d}$ is a symmetric matrix.

**Lemma 3.2.** *Suppose* $\mathbf{A}$ *has the eigenvalue decomposition as* $\mathbf{A} = \sum_{j=1}^{d} \sigma_j \boldsymbol{v}_j \boldsymbol{v}_j^T$, *where* $\sigma_j$'s *are the eigenvalues and* $\boldsymbol{v}_j$'s *are corresponding eigenvectors. Let* $\widetilde{\sigma}_j = \max\{\sigma_j, 0\}$ *for* $j = 1, ..., d$, *then we have* $\Pi_+(\mathbf{A}) = \sum_{j=1}^{d} \widetilde{\sigma}_j \boldsymbol{v}_j \boldsymbol{v}_j^T$.

Now we start with the $t$-th iteration. We first calculate the auxiliary variable $\mathbf{M}^{(t)}$ as

$$\mathbf{M}^{(t)} = (1 - \theta_t)\mathbf{S}^{(t-1)} + \theta_t \mathbf{W}^{(t-1)}. \tag{3.9}$$

We then evaluate the gradient according to (3.6),

$$\mathbf{G}^{(t)} = \frac{\partial \|\widehat{\mathbf{S}} - \mathbf{M}^{(t)}\|_\infty^\mu}{\partial \mathbf{M}^{(t)}}. \tag{3.10}$$

We consider the following quadratic approximation

$$Q(\mathbf{W}, \mathbf{W}^{(t-1)}, \mu) = \|\widehat{\mathbf{S}} - \mathbf{W}^{(t-1)}\|_\infty^\mu + \left\langle \mathbf{G}^{(t)}, \mathbf{W} - \mathbf{W}^{(t-1)} \right\rangle + \frac{1}{2\mu\theta_t} \|\mathbf{W} - \mathbf{W}^{(t)}\|_\mathsf{F}^2. \tag{3.11}$$

By simple manipulations and Lemma 3.2, the fast proximal gradient algorithm takes

$$\mathbf{W}^{(t)} = \underset{\mathbf{W} \succeq 0}{\operatorname{argmin}}\, Q(\mathbf{W}, \mathbf{W}^{(t-1)}, \mu) = \Pi_+\left( \mathbf{W}^{(t-1)} - \frac{\mu}{\theta_t} \mathbf{G}^{(t)} \right), \tag{3.12}$$

where $\mu$ works as a step-size here. We further calculate $\mathbf{S}^{(t)}$ for the $t$-th iteration as follow,

$$\mathbf{S}^{(t)} = (1 - \theta_t)\mathbf{S}^{(t-1)} + \theta_t \mathbf{W}^{(t)}. \tag{3.13}$$

**Theorem 3.3.** *Given the desired accuracy* $\epsilon$ *such that* $\|\widehat{\mathbf{S}} - \mathbf{S}^{(t)}\|_\infty^\mu - \|\widehat{\mathbf{S}} - \widetilde{\mathbf{S}}\|_\infty^\mu < \epsilon$, *we need the number of iterations to be at most*

$$t = \sqrt{2\|\mathbf{S}^{(0)} - \widetilde{\mathbf{S}}\|_\mathsf{F}^2 / (\mu\epsilon)} - 1 = O\left( \sqrt{1/(\mu\epsilon)} \right). \tag{3.14}$$

The detailed proof can be found in the extended draft Zhao et al. (2013) due to the space limit. Theorem 3.3 guarantees that our derived algorithm achieves the optimal rate of convergence for minimizing (3.7) over the class of all gradient-based computational algorithms. In next section, by directly analyzing the tradeoff between the computational efficiency and statistical error, we will show that choosing a suitable smooth parameter $\mu$ allows $\widetilde{\mathbf{S}}$ concentrate to $\boldsymbol{\Sigma}^*$ with a rate similar to Lemma 2.2 in high dimensions, though (3.7) is not the same as the original projection problem (3.2).

## 4 Statistical Properties

In this section we present the statistical properties of the proposed method. Due to space limit, all the proofs of following theorems can be found in the extended draft Zhao et al. (2013). The next theorem establishes the concentration property of $\widetilde{\mathbf{S}}$ under the elementwise $\ell_\infty$ norm. This result will be useful to prove later main theorem.

**Theorem 4.1.** *Given the nonparanormal* SKEPTIC *estimator* $\widehat{\mathbf{S}}$, *for any large enough* $n$, *under the conditions that* $\mu \leq 4\pi\varphi$ *and* $\varphi > 0$, *we have the optimum to* (3.7), *denoted as* $\widetilde{\mathbf{S}}$, *satisfying*

$$\mathbb{P}\left( \|\widetilde{\mathbf{S}} - \boldsymbol{\Sigma}^*\|_\infty \leq 18\pi\varphi \right) \geq 1 - d^2 \exp(-n\varphi^2). \tag{4.1}$$

Theorem 4.1 is non-asymptotic. It implies that we can gain the computational efficiency without losing statistical rate in terms of elementwise sup-norm as long as $\mu$ is reasonably large. We now show that our proposed smooth-projected neighborhood approach recovers the true neighborhood for each node with high probability under the following irrepresentable condition (Zhao and Yu, 2006; Zou, 2006; Wainwright, 2009).

**Assumption 1** (Irrepresentable Condition). *Recall that* $I_j$ *and* $J_j$ *denote the true neighborhood and non-neighborhood of node* $j$ *respectively. There exist* $\alpha \in (0, 1)$, $\delta_{\min} > 0$ *and* $\delta^{-1} \leq \psi < \infty$ *such that for* $\forall j = 1, .., d$, *the following conditions hold,*

$$(C.1)\ \|\boldsymbol{\Sigma}_{J_j I_j}^* (\boldsymbol{\Sigma}_{I_j I_j}^*)^{-1}\|_\infty \leq \alpha; \tag{4.2}$$

$$(C.2)\ \Lambda_{\min}(\boldsymbol{\Sigma}_{I_j I_j}^*) \geq \delta,\ \ \|(\boldsymbol{\Sigma}_{I_j I_j}^*)^{-1}\|_\infty \leq \psi. \tag{4.3}$$

The proposed projection approach can be also combined with other graph estimation method such as Zhou et al. (2009), in which the conditions above can be relaxed. Here we use this condition for an illustrative purpose to show that the proposed method has a theoretical guarantee.

**Theorem 4.2** (Graph Recovery Performance). *Let $\tau = \min |\mathbf{B}_{jk}^*|$ for all $(j,k)$'s such that $\mathcal{G}_{jk}^* \neq 0$, where $\mathbf{B}^* \in \mathbb{R}^{d \times d}$ with $\mathbf{B}_{\backslash j,j}^* = (\mathbf{\Sigma}_{\backslash j,\backslash j}^*)^{-1} \mathbf{\Sigma}_{\backslash j,j}^*$ and $\mathbf{B}_{j,j}^* = 0$. We assume that $\mathbf{\Sigma}^*$ satisfies Conditions C.1 and C.2. Let $s_j = |I_j| < n$ and choose the $\lambda$ such that $\lambda \leq \min \{\tau/\psi, 2\}$, then there exist positive universal constants $c_0$ and $c_1$, such that*

$$
\mathbb{P}\left(\widehat{J}_j = J_j, \widehat{I}_j = I_j\right) \geq 1 - s_j^2 \exp\left(\frac{-c_1 n \delta^2}{4 s_j^2}\right) - s_j^2 \exp\left(-\frac{c_1 n \varphi^2}{s_j^2}\right)
$$

$$
- (d - s_j) s_j \exp\left(-\frac{c_1 n \varphi^2}{s_j^2}\right) - d \exp(-c_1 n \varphi^2), \quad (4.4)
$$

*where $\varphi$ satisfies that $c_0 \sqrt{\frac{\log d}{n}} \leq \varphi \leq \min \left\{1, \frac{1}{2\psi}, \frac{\lambda(1-\alpha)}{26\psi}, \frac{\lambda(1-\alpha)}{26(\alpha+1)}, \frac{\tau}{14\psi^2}, \frac{\tau}{14\psi}\right\}$.*

Theorem 4.2 is also non-asymptotic. It guarantees that for each individual node, we can correctly recover its neighborhood with high probability. Consequently, the following corollary can be implied so that we can asymptotically recover the underlying graph structure under given conditions.

**Corollary 4.3.** *Let $s = \max_{1 \leq j \leq d} s_j$, then under the same conditions as in Theorem 4.2, we have $\mathbb{P}(\widehat{\mathcal{G}} = \mathcal{G}) \to 1$ if the following conditions hold:*

*(C.3) $\alpha$, $\delta$ and $\psi$ are constants, which do not scale with the triplet $(n, d, s)$;*

*(C.4) The triplet $(n, d, s)$ scales as $s^2(\log d + \log s)/n \to 0$ and $s^2 \log d/(\tau^2 n) \to 0$;*

*(C.5) $\lambda$ scales with $(n, d, s)$ as $\lambda/\tau \to 0$ and $s^2 \log d/(\lambda^2 n) \to 0$.*

## 5  Numerical Simulations

Liu et al. (2012) recommend to use the Kendall's tau for nonparanormal graph estimation because of its superior robustness property compared to the Spearman's rho. In this section, we use the Kendall's tau in our smooth-projected neighborhood pursuit method. For synthetic data, we use the following four different graphs with 200 nodes ($d = 200$): (1) Erdös-Rényi graph; (ii) Cluster graph; (iii) Chain graph; (4) Scale-free graph. We simulate data from the Gaussian distributions that Markov to the above graphs. We adopt the power function $g(t) = \text{sign}(t)|t|^4$ to convert the Gaussian data to the nonparanormal data. More details about the data simulation can be found in Zhao et al. (2013). We use the ROC curve to evaluate the graph estimation performance. Since $d > n$, the full solution paths cannot be obtained, therefore we restrict the range of false positive edge discovery rates to be from 0 to 0.3 for computational convenience.

### 5.1  Our proposed method vs. Nonparanormal SKEPTIC Estimator

We first evaluate the proposed smoothed elementwise $\ell_\infty$-norm projection algorithm. For this, we sampled 100 data points from a 200-dimensional standard normal distribution $N(\mathbf{0}, \mathbf{I}_{200})$. We study the empirical performance of the proposed fast proximal gradient algorithm using different smoothing parameters ($\mu = 1, 0.5, 0.25, 0.1$). The optimization and statistical error curves for different smoothing parameters (averaged over 50 replications) are presented in Figure 1. Figure 1(a) shows the original objective value $\|\widehat{\mathbf{S}} - \mathbf{S}^{(t)}\|_\infty$ v.s. the number of iterations. Compared with smaller $\mu$'s, we see that choosing $\mu = 1$ reduces the computational burden but increases the approximation error w.r.t the problem in (3.2). However, Figure 1(b) shows that, in terms of the statistical error $\|\mathbf{\Sigma}^* - \mathbf{S}^{(t)}\|_\infty$, $\mu = 1$ performs similarly to the other smaller $\mu$'s. Therefore, we show that significant computational efficiency can be gained with little loss of statistical error.

We further compare the graph recovery performance of our proposed method with the naive indefinite nonparanormal SKEPTIC estimator as in Liu et al. (2012). The averaged ROC curves over 100 replications are presented in Figure 2. We see that directly plugging the indefinite nonparanormal SKEPTIC estimator into the neighborhood pursuit results in the worst performance. The ROC performance drops dramatically due to the non-convexity of the objective function. While

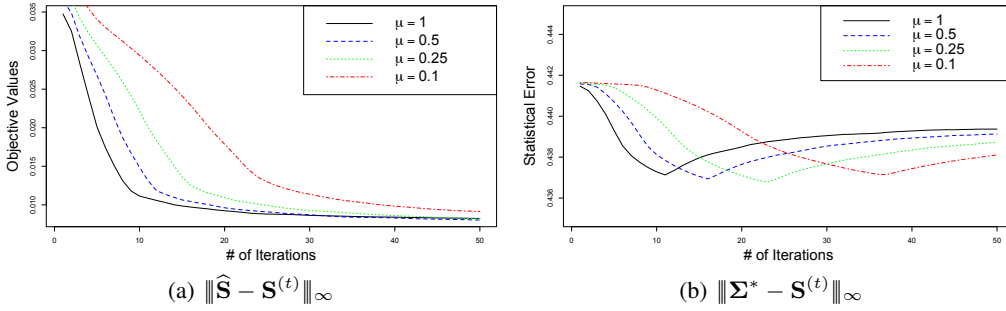

(a) $\|\|\widehat{\mathbf{S}} - \mathbf{S}^{(t)}\|\|_\infty$  (b) $\|\|\mathbf{\Sigma}^* - \mathbf{S}^{(t)}\|\|_\infty$

Figure 1: The empirical performance using different smoothing parameters. $\mu = 1$ has a similar performance to the smaller $\mu$'s in terms of the estimation error.

our smoothed-projected neighborhood pursuit method significantly outperforms the naive indefinite nonparanormal SKEPTIC estimator.

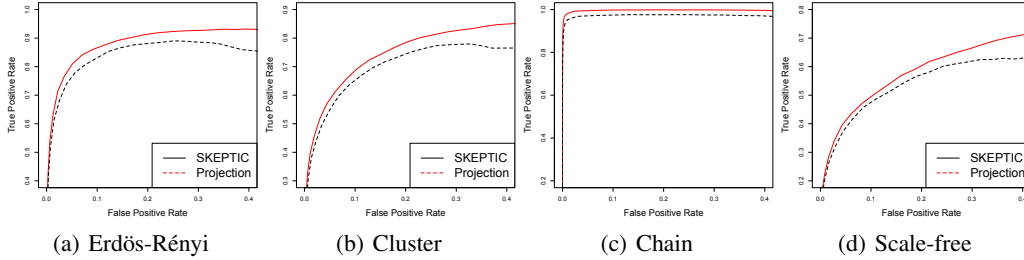

(a) Erdös-Rényi  (b) Cluster  (c) Chain  (d) Scale-free

Figure 2: The averaged ROC curves of the neighborhood pursuit when combined with different correlation estimators. "SKEPTIC" represents the indefinite nonparanormal SKEPTIC estimator, and "Projection" represents our proposed projection approach.

## 5.2 Our Proposed Method vs. Naive Neighborhood Pursuit

In this subsection, we conduct similar numerical studies as in Liu et al. (2012) to compare our proposed method with the naive neighborhood pursuit method. The naive neighborhood pursuit directly exploits the Pearson correlation estimator under the neighborhood pursuit framework. Choosing $n = 100$ and $d = 200$, we use the same experimental setup as in the previous subsection. The averaged ROC curves over 100 replications are presented in Figure 3. As can be seen, our proposed projection method outperforms the naive neighborhood pursuit throughout all four different graphs.

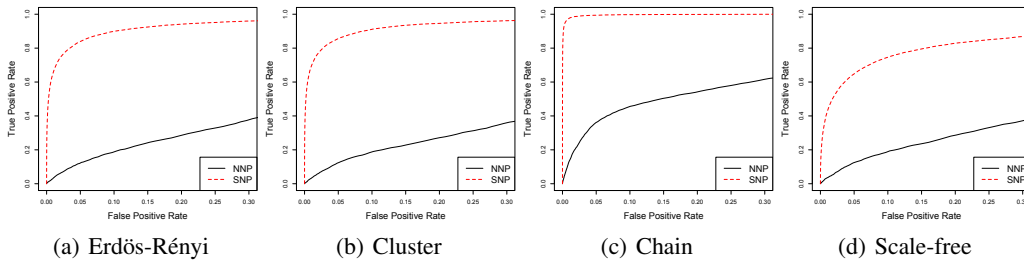

(a) Erdös-Rényi  (b) Cluster  (c) Chain  (d) Scale-free

Figure 3: The averaged ROC curves of the neighborhood pursuit when combined with different correlation estimators. "SNP" represents our proposed estimator and "NNP" represents the Pearson estimator. The SNP uniformly outperforms the NNP for all four graphs.

## 6  Real Data Analysis

In this section we present a real data experiment to compare the nonparanormal graphical model to Gaussian graphical model. For model selection, we use the stability graph procedure (Meinshausen

and Bühlmann, 2010; Liu et al., 2010), which has the following steps: (1) Calculate the solution path using all the samples, and choose the regularization parameter at the sparsity level $4\%$; (2) Randomly choose $10\%$ of all the samples without replacement using the regularization parameter chosen in (1); (3) Repeat the step (2) 500 times and retain the edges that appear with frequencies no less than $95\%$.

The topic graph is first used in Blei and Lafferty (2007) to illustrate the idea of correlated topic modeling. The correlated topic model, is a hierarchical Bayesian model for abstracting $K$ "topics" that occur in a collection of documents (corpus). By applying the variational EM-algorithm, we can estimate the topic proportion for each document and represent it in a $K$-dimensional simplex (mixed-membership). Blei and Lafferty (2007) assume that the topic proportion approximately follows a normal distribution after the logarithmic-transformation. Here we are interested in visualizing the relationship among the topics using an undirected graph: the nodes represent individual topics, and edges connecting different nodes represent highly related topics. The corpus used in Blei and Lafferty (2007) contains 16,351 documents with 19,088 unique terms. Similar to Blei and Lafferty (2007), we choose $K = 100$ and fit a topic model to the articles published in *Science* from 1990 to 1999.

Evaluated by the Kolmogorov-Smirnov test, we find many topic data highly violate the normality assumption (More details can be found in Zhao et al. (2013)). This motivates our choice of the smooth-projected neighborhood pursuit approach. The estimated topic graphs are provided in Figure 4. The smooth-projected neighborhood pursuit generates 6 mid-size modules and 6 small modules, while the naive neighborhood pursuit generated 1 large module, 2 mid-size modules and 6 small modules. The nonparanormal approach discovers more refined structures and improves the interpretability of the obtained graph. For example: (1) Topics closely related to climate change in Antarctica are clustered in the same module such as "ice-68", "ozone-23" and "carbon-64"; (2) Topics closely related to environmental ecology are clustered in the same module such as "monkey-21", "science-4", "environmental-67", "species-86", etc.; (3) Topics closely related to modern physics are clustered in the same module such as "quantum-29", "magnetic-55", "pressure-92", "solar-62", etc.. In contrast, the naive neighborhood pursuit mixes all these different topics in a large module.

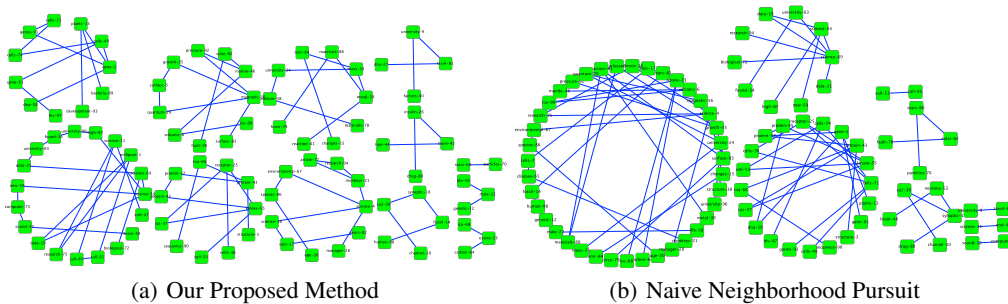

(a) Our Proposed Method          (b) Naive Neighborhood Pursuit

Figure 4: Two topic graphs illustrating the difference of the estimated topic graphs. The smooth-projected neighborhood pursuit (subfigure (a)) generates 6 mid-size modules and 6 small modules while the naive neighborhood pursuit (subfigure (b)) generates 1 large module, 2 mid-size modules and 6 small modules.

## 7 Conclusion and Acknowledgement

In this paper, we study how to estimate the nonparanormal graph using the neighborhood pursuit in conjunction with the possible indefinite nonparanormal skeptic estimator. Using our proposed smoothed projection approach, the resulting estimator can be used as a positive semi-definite refinement of the nonparanormal skeptic estimator. Our estimator has better graph estimation performance with theoretical guarantee. Our results suggest that it is possible to gain estimation robustness and modeling flexibility without losing two important computational structures: convexity and smoothness. The topic modeling experiment demonstrates that our proposed method may lead to more refined scientific discovery. Han Liu and Tuo Zhao are supported by NSF award IIS-11167308, and Kathryn Roeder is supported by National Institute of Mental Health grant MH057881.

# References

BANERJEE, O., GHAOUI, L. E. and D'ASPREMONT, A. (2008). Model selection through sparse maximum likelihood estimation. *Journal of Machine Learning Research* **9** 485–516.

BLEI, D. and LAFFERTY, J. (2007). A correlated topic model of science. *Annals of Applied Statistics* **1** 17–35.

CHEN, X., LIN, Q., KIM, S., CARBONELL, J. and XING, E. (2012). A smoothing proximal gradient method for general structured sparse regression. *Annals of Applied Statistics* To appear.

DEMPSTER, A. (1972). Covariance selection. *Biometrics* **28** 157–175.

FRIEDMAN, J., T. HASTIE, H. H. and TIBSHIRANI, R. (2007). Pathwise coordinate optimization. *Annals of Applied Statistics* **1** 302–332.

HONORIO, J., ORTIZ, L., SAMARAS, D., PARAGIOS, N., and GOLDSTEIN, R. (2009). Sparse and locally constant gaussian graphical models. *Advances in Neural Information Processing Systems* 745–753.

KLAASSEN, C. and WELLNER, J. (1997). Efficient estimation in the bivariate normal copula model: Normal margins are least-favorable. *Bernoulli* **3** 55–77.

LAURITZEN, S. (1996). *Graphical models*, vol. 17. Oxford University Press, USA.

LIU, H., HAN, F., YUAN, M., LAFFERTY, J. and WASSERMAN, L. (2012). High dimensional semiparametric gaussian copula graphical models. *Annals of Statistics* To appear.

LIU, H., LAFFERTY, J. and WASSERMAN, L. (2009). The nonparanormal: Semiparametric estimation of high dimensional undirected graphs. *Journal of Machine Learning Research* **10** 2295–2328.

LIU, H., ROEDER, K. and WASSERMAN, L. (2010). Stability approach to regularization selection for high dimensional graphical models. *Advances in Neural Information Processing Systems* .

MEINSHAUSEN, N. and BÜHLMANN, P. (2006). High dimensional graphs and variable selection with the lasso. *Annals of Statistics* **34** 1436–1462.

MEINSHAUSEN, N. and BÜHLMANN, P. (2010). Stability selection. *Journal of the Royal Statistical Society, Series B* **72** 417–473.

NESTEROV, Y. (1988). On an approach to the construction of optimal methods of smooth convex functions. *Ékonom. i. Mat. Metody* **24** 509–517.

NESTEROV, Y. (2005). Smooth minimization of non-smooth functions. *Mathematical Programming* **103** 127–152.

RAVIKUMAR, P., LAFFERTY, J., LIU, H. and WASSERMAN, L. (2009). Sparse additive models. *Journal of the Royal Statistical Society, Series B* **71** 1009–1030.

RAVIKUMAR, P., WAINWRIGHT, M., RASKUTTI, G. and YU, B. (2011). High-dimensional covariance estimation by minimizing $\ell_1$-penalized log-determinant divergence. *Electronic Journal of Statistics* **5** 935–980.

TSUKAHARA, H. (2005). Semiparametric estimation in copula models. *Canadian Journal of Statistics* **33** 357–375.

WAINWRIGHT, M. (2009). Sharp thresholds for highdimensional and noisy sparsity recovery using $\ell_1$ constrained quadratic programming. *IEEE Transactions on Information Theory* **55** 2183–2201.

WILLE, A., ZIMMERMANN, P., VRANOVA, E., FRHOLZ, A., LAULE, O., BLEULER, S., HENNIG, L., PRELIC, A., VON ROHR, P., THIELE, L., ZITZLER, E., GRUISSEM, W. and BÜHLMANN, P. (2004). Sparse graphical gaussian modeling of the isoprenoid gene network in arabidopsis thaliana. *Genome Biology* **5** R92.

YUAN, M. and LIN, Y. (2007). Model selection and estimation in the gaussian graphical model. *Biometrika* **94** 19–35.

ZHAO, P. and YU, B. (2006). On model selection consistency of lasso. *Journal of Machine Learning Research* **7** 2541–2563.

ZHAO, T., LIU, H., ROEDER, K., LAFFERTY, J. and WASSERMAN, L. (2012). The huge package for high-dimensional undirected graph estimation in r. *Journal of Machine Learning Research* To appear.

ZHAO, T., ROEDER, K. and LIU, H. (2013). A smoothing projection approach for high dimensional nonparanormal graph estimation. Tech. rep., Johns Hopkins University.

ZHOU, S., VAN DE GEER, S. and BÜHLMANN, P. (2009). Adaptive lasso for high dimensional regression and gaussian graphical modeling. Tech. rep., ETH Zurich.

ZOU, H. (2006). The adaptive lasso and its oracle properties. *Journal of the American Statistical Association* **101** 1418–1429.

